# Max-margin classification of incomplete data

**Gal Chechik[1], Geremy Heitz[2],**
**Gal Elidan[1], Pieter Abbeel [1], Daphne Koller [1]**
[1] Department of Computer Science, Stanford University, Stanford CA, 94305
[2] Department of Electrical Engineering, Stanford University, Stanford CA, 94305
`Email for correspondence: gal@ai.stanford.edu`

## Abstract

We consider the problem of learning classifiers for structurally incomplete data, where some objects have a subset of features inherently absent due to complex relationships between the features. The common approach for handling missing features is to begin with a preprocessing phase that completes the missing features, and then use a standard classification procedure. In this paper we show how incomplete data can be classified *directly* without any completion of the missing features using a max-margin learning framework. We formulate this task using a geometrically-inspired objective function, and discuss two optimization approaches: The linearly separable case is written as a set of convex feasibility problems, and the non-separable case has a non-convex objective that we optimize iteratively. By avoiding the pre-processing phase in which the data is completed, these approaches offer considerable computational savings. More importantly, we show that by elegantly handling complex patterns of missing values, our approach is both competitive with other methods when the values are missing at random and outperforms them when the missing values have non-trivial structure. We demonstrate our results on two real-world problems: edge prediction in metabolic pathways, and automobile detection in natural images.

## 1 Introduction

In the traditional formulation of supervised learning, data instances are viewed as vectors of features in some high-dimensional space. However, in many real-world tasks, data instances have a complex pattern of missing features. While features may sometimes be missing due to measurement noise or corruption, different samples often have different sets of observable features due to *inherent* properties of the instances. For example, in the case of recognizing objects in natural images, an object is often classified using a set of image patches corresponding to parts of the object (like the license plate for cars); but some images may not contain all parts, either because a part was not captured in the image or because the specific instance does not have this part in the first place.

In other scenarios, some features cannot even be defined for all instances. Such situations arise when the objects to be learned are organized based on a known graph structure, since their features may rely on local properties of the graph. For example, we might wish to classify the attributes of a web-page given the attributes of neighboring web-pages [8]. In analyzing genomic data, we may wish to predict the edges in networks of interacting proteins or chemical reactions [9, 15]. In these cases, the local neighborhood of an instance in the graph often varies drastically, and it has already been observed that variation this could introduce statistical biases [8]. In the web-page task, for instance, a useful feature is the most common topic of other sites that point to a given page. When a page has no such parents, however, this feature is meaningless and should be considered *structurally absent*.

The common approach for classification with missing features is *imputation*, a two phase procedure where the values of the missing attributes are first filled in during a preprocessing

phase, after which a standard classifier is applied to the completed data [10]. Most Imputation techniques make most sense when the features are missing due to noise, especially in the setting of *missing at random* (MAR, when the missingness pattern is conditionally independent of the unobserved features given the observations), or *missing completely at random* (MCAR, when it is independent of both observed and unobserved measurements).

In common practice of applying imputation, missing attributes in continuous data are often filled with zeros, or with the average of all of the data instances, or using the *k nearest neighbors* (kNN) of each instance to find a plausible value of its missing features. A second family of imputation methods builds probabilistic generative models of the features using raw maximum likelihood or algorithms such as expectation maximization (EM) [4]. Such model-based methods allow the designer to introduce prior knowledge and are extremely useful when priors can be explicitly modeled. These methods work very well for MAR data settings, because they assume that the missing features are generated by the same model that generates the observed features. However, model-based approaches can be computationally expensive, and require significant prior knowledge about the data. More importantly, they will produce meaningless completions for features that are *structurally absent*. As an extreme example, consider two subpopulation of instances (e.g., animals and buildings) having no overlapping features (e.g., body parts, and architectural aspects), in which filling missing values (e.g., the body parts of buildings) is clearly meaningless and may harm classification performance. As a result, for structurally absent features, it would be useful if we could avoid unnecessary prediction of hypothetical undefined values, and classify instances directly.

We approach this problem directly from the geometric interpretation of the classification task as finding a separating hyperplane in the feature space. We view instances with different feature sets as lying in subspaces of the full feature space, and suggest a modified optimization objective within the framework of support vector machines (SVMs), that explicitly considers the subspace of each instance. We show how the linearly separable case can be efficiently solved using convex optimization (*second order cone programming*, SOCP). The objective of the non separable case is non-convex, and we propose an iterative procedure that is found to converge in practice. These approaches may be viewed as model-free methods for handling missing data in the cases where the MAR assumption fails to hold.

We evaluate the performance of our approach in two real world applications: prediction of missing enzymes in a metabolic network, and automobile detection in natural images. In both tasks, features may be inherently absent due to the mechanisms described above, and our methods are found superior to other simple imputation methods.

## 2 Max-Margin Formulation for Missing Features

Let $\mathbf{x}_1 \dots \mathbf{x}_n$ be a set of samples with binary labels $y_i \in \{-1, 1\}$. Each sample $x_i$ is characterized by a subset of features $\mathcal{F}(\mathbf{x}_i)$, from a full set $\mathcal{F}$ of size $d$. A sample that has all features $\mathcal{F}(\mathbf{x}_i) = \mathcal{F}$, is viewed as a vector in $\mathbb{R}^d$, where the $i^{th}$ coordinate corresponds to the $i^{th}$ feature. A sample $\mathbf{x}_i$ with partially valid features can be viewed as embedded in the relevant subspace $\mathbb{R}^{|\mathcal{F}(\mathbf{x}_i)|} \subseteq \mathbb{R}^d$. For simplicity of notation, we treat each $\mathbf{x}_i$ as if it were a vector in $\mathbb{R}^d$ where only its $\mathcal{F}(\mathbf{x}_i)$ entries are valid and define the inner product with another vector in $\mathbb{R}^d$ as $\mathbf{wx} = \sum_{k:f_k \in \mathcal{F}(\mathbf{x}_i)} w_k x_k$. Importantly, since instances share features, the learned classifier must be consistent across instances, assigning the same weight to a given feature in different samples, even if those instance do not lie in the same subspace.

In the classical SVM approach [14, 13], a linear classifier $\mathbf{w}$ is optimized to maximize the margin, defined as $\min_i y_i(\mathbf{wx}_i + b)/\|\mathbf{w}\|$, and the learning problem is reduced to the quadratic constrained optimization problem

$$\min_{\mathbf{w},\xi,b} \quad \frac{1}{2}\|\mathbf{w}\|^2 + C\sum_{i=1}^{n} \xi_i \qquad s.t. \quad y_i(\mathbf{wx}_i + b) \geq 1 - \xi_i , \quad i = 1 \dots n \qquad (1)$$

where $b$ is a threshold, the $\xi$'s are slack variables necessary for the case when the training instances are not linearly separable, and $C$ is the error penalty. Eq. (1) can be extended to nonlinear classifiers using kernels [13].

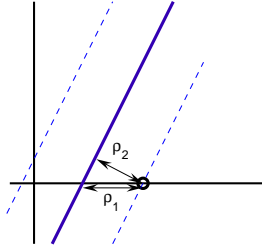

Figure 1: The margin is incorrectly scaled when a sample that has missing features is treated as if the missing features have a value of zero. In this example, the margin of a sample that only has one feature (the $x$ dimension) is measured both in the higher dimensional space ($\rho_2$) and the lower one ($\rho_1$). If all features are assumed to exist, and we give the missing feature (along the $y$ axis) a value of zero, the margin $\rho_2$ measured in the higher dimensional space is shorter that the margin measured in the relevant subspace $\rho_1$ .

Consider now learning such a classifier in the presence of missing data. At first glance, it may appear that since the $\mathbf{x}$'s only affect the optimization through inner products with $\mathbf{w}$, missing features can merely be skipped (or equivalently, replaced with zeros), thereby preserving the values of the inner product. However, this does not properly normalize the different entries in $\mathbf{w}$, and damages classification accuracy. The reason is illustrated in Fig. 1 where a single sample in $\mathbb{R}^2$ has one valid and one missing feature. Due to the missing feature, measuring the margin in the full space $\rho_2$, underestimates the correct geometric margin of the sample in the valid space $\rho_1$. This is different from the case where the feature exists but is unknown, in which the sample's margin could be either over- or under-estimated. In the next sections, we explore how this Eq. (1) can be solved while properly taking this normalization into account. We start by reminding the reader about the geometric interpretation of SVM.

## 3   Geometric interpretation

The derivation of the SVM classifier [14] is motivated by the goal of finding a hyperplane that maximally separates the positive examples from the negative, as measured by the geometric margin $\rho(\mathbf{w}) = \min_i \frac{y_i \mathbf{w} \mathbf{x}_i}{\|\mathbf{w}\|}$. The task of maximizing the margin $\rho(\mathbf{w})$,

$$\max_{\mathbf{w}} \rho(\mathbf{w}) = \max_{\mathbf{w}} \left( \min_i \frac{y_i \mathbf{w} \mathbf{x}_i}{\|\mathbf{w}\|} \right) \tag{2}$$

is transformed into the quadratic programming problem of Eq. (1) in two steps. First, $\|\mathbf{w}\|$, is taken out of the minimization, yielding $\max_{\mathbf{w}} \frac{1}{\|\mathbf{w}\|} (\min_i y_i \mathbf{w} \mathbf{x}_i)$. Then, the following invariance is used: for every solution, there exists a solution that achieves the same target function value, but with a margin that equals 1. This allows us to write the SVM problem as a constrained optimization problem: $\max_{\mathbf{w}} \|\mathbf{w}\|^{-1}$ s.t. $y_i(\mathbf{w}\mathbf{x}_i) \geq 1$. This is equivalent to minimizing $\|\mathbf{w}\|^2$ with the same constraints, which equals the SVM problem of Eq. (1).

In the case of missing features, this derivation no longer optimizes the correct geometrical margin (Fig. 1). To address this problem, we treat the margin of each instance in its own subspace, by defining the *instance margin* for the $i^{th}$ instance as $\rho_i(\mathbf{w}) = \frac{y_i \mathbf{w}^{(i)} \mathbf{x}_i}{\|\mathbf{w}^{(i)}\|}$ where $\|\mathbf{w}^{(i)}\| = \sqrt{\sum_{k: f_k \in \mathcal{F}(\mathbf{x}_i)} w_k^2}$. The geometric margin is, as before, the minimum over all instance margins, yielding a new optimization problem

$$\max_{\mathbf{w}} \left( \min_i \frac{y_i \mathbf{w}^{(i)} \mathbf{x}_i}{\|\mathbf{w}^{(i)}\|} \right). \tag{3}$$

Unfortunately, since different margin terms are normalized by different norms $\|\mathbf{w}^{(i)}\|$, we can no longer take the denominator out of the minimization as above. In addition, each of the terms $y_i \mathbf{w}^{(i)} \mathbf{x}_i / \|\mathbf{w}^{(i)}\|$ is non-convex in $\mathbf{w}$, which is difficult to solve directly in an efficient way. We now discuss two approaches for solving this problem.

In the linearly separable case, the optimization problem of Eq. (3) is equivalent to

$$\max_{\mathbf{w}, \gamma} \quad \gamma \quad s.t. \quad y_i \mathbf{w}^{(i)} \mathbf{x}_i \geq \gamma \|\mathbf{w}^{(i)}\| \quad i = 1 \dots n \quad , \tag{4}$$

This is a convex feasibility problem for any fixed value of $\gamma$, which is a real scalar that corresponds to the margin. It can be solved efficiently using a bisection search over $\gamma \in \mathbb{R}^+$, where in each iteration we solve a convex *second order cone program* (SOCP) [11]. Unfortunately, extending this formulation to the non-separable while preserving the geometric margin interpretation case makes the problem non-convex (this is discussed elsewhere).

A second approach for solving Eq. (3) is to treat each instance margin individually. We represent each of the norms $\|\mathbf{w}^{(i)}\|$ as a scaling of the full norm by defining scaling coefficients $s_i = \|\mathbf{w}^{(i)}\|/\|\mathbf{w}\|$, and rewriting Eq. (3) as

$$\max_{\mathbf{w}} \left( \min_i \frac{y_i \mathbf{w} \mathbf{x}_i}{s_i \|\mathbf{w}\|} \right) = \max_{\mathbf{w}} \frac{1}{\|\mathbf{w}\|} \left( \min_i \frac{y_i \mathbf{w} \mathbf{x}_i}{s_i} \right) \ , \quad s_i = \frac{\|\mathbf{w}^{(i)}\|}{\|\mathbf{w}\|}. \tag{5}$$

The $s_i$ factors are scalars, and had we known them, we could have solved a standard SVM problem. Unfortunately they depend on $\mathbf{w}^{(i)}$ and are unknown. This formalism allows us to use again the invariance to the rescaling of $\|\mathbf{w}\|$ and rewrite as a constrained optimization problem over $s_i$ and $\mathbf{w}$. In the non-separable case, Eq. (5) becomes

$$\min_{\mathbf{w},b,\xi,\mathbf{s}} \quad \frac{1}{2}\|\mathbf{w}\|^2 + C \sum_i \xi_i \qquad s.t. \quad \frac{1}{s_i} \left( y_i(\mathbf{w}\mathbf{x}_i + b) \right) \geq 1 - \xi_i \ , \quad i = 1 \ldots n \tag{6}$$

$$s_i = \|\mathbf{w}^{(i)}\|/\|\mathbf{w}\| \ , \quad i = 1 \ldots n$$

This constrained optimization problem is no longer a QP. In fact, due to the normalization constraint it is not even convex in $\mathbf{w}$. One solution is a *projected gradient* approach, in which one iterates between steps in the direction of the gradient of the Lagrangian and projections to the constrained space, by calculating $s_i = \|\mathbf{w}^{(i)}\|/\|\mathbf{w}\|$. For the right choices of step sizes, such approaches are guaranteed to converge to local minima [2].

We can use a faster iterative algorithm based on the fact that the problem is a QP for any given set of $s_i$'s, and iterate between (1) solve a QP for $\mathbf{w}$ given $s_i$, and (2) use the resulting $\mathbf{w}$ to calculate new $s_i$'s. This algorithm differs from a projected gradient approach in that rather than taking a series of small gradient steps, it takes bigger leaps, and projects back to the constrained space after each step. Since the convergence of this iterative algorithm is not guaranteed, we used cross validation to choose an early stopping point and found that the best solutions were obtained within 2-5 steps. Typically, the objective improved on the first 1-3 iterations, but then, in about 75% of the cases the objective oscillated. In the remaining cases the algorithm converged to a fixed point. It is easy to see that a fixed point of this iterative procedure achieves an optimal solution for Eq. (6), since it achieves a minimal $\|\mathbf{w}\|$ while obeying the $s_i$ constraints. As a result, when this algorithm converges, the solution is also guaranteed to be a locally optimal solution of the original problem Eq. (3).

The power of the SVM approach can be largely attributed to the flexibility and efficiency of nonlinear classification allowed through the use of kernels. The dual of the above QP can be kernelized as in a standard SVM, yielding

$$\max_{\alpha \in \mathbb{R}^n} \sum_{i=1}^{n} \alpha_i - \frac{1}{2} \sum_{i,j=1}^{n} \alpha_i \frac{y_i}{s_i} K\left(\mathbf{x}_i, \mathbf{x}_j\right) \frac{y_j}{s_j} \alpha_j \quad s.t. \quad 0 \leq \alpha_i \leq C \ ; \quad \sum_{i=1}^{n} \alpha_i y_i = 0. \tag{7}$$

where $K(\mathbf{x}_i, \mathbf{x}_j)$ is the kernel function that simulates an inner product in the higher dimensional feature space. Classification of new samples are obtained as in standard SVM by calculating the margin $\rho(x_{new}) = \sum_j y_j \alpha_j \frac{1}{s_j} K(x_j, x_{new}) \frac{1}{s_{new}}$.

Kernels in this formulation operate over vectors with missing features, hence we have to develop kernels that handle them correctly. Fortunately, many kernels only depend on their inputs through their inner product. In this case there is an easy procedure to construct a modified kernel that takes such missing values into account. For example, for a polynomial kernel $K(\mathbf{x}_i, \mathbf{x}_j) = (\langle \mathbf{x}_i, \mathbf{x}_j \rangle + 1)^d$, define $K'(\mathbf{x}_i, \mathbf{x}_j) = K(\mathbf{x}_i, \mathbf{x}_j) = (\langle \mathbf{x}_i, \mathbf{x}_j \rangle_{\mathcal{F}} + 1)^d$, with the inner product calculated over valid features $\langle \mathbf{x}_i, \mathbf{x}_j \rangle_{\mathcal{F}} = \sum_{k: f_k \in \chi(\mathbf{x}_j) \cap \mathcal{F}(\mathbf{x}_i)} \langle \mathbf{x}_{ik}, \mathbf{x}_{jk} \rangle$. This can be easily proved to be a kernel.

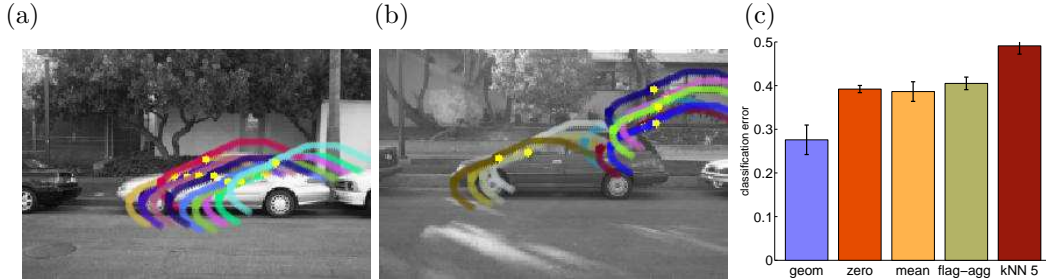

Figure 2: Car classification results. **(a)** An easy instance where all local features are approximately in agreement. **(b)** A hard instance where local features are divided into two distinct groups. This instance was correctly classified by the 'geometric margin' approach but misclassified by all other methods. **(c)** Classification accuracy of the different methods for the task of object recognition in real images. Error bars are standard errors of the mean (SEM) over the five cross validation sets.

## 4 Experiments

We evaluated our approaches in three different missingness scenarios. First, as a sanity check, we explored performance when features are missing at random, in a series of five standard UCI benchmarks, and also in a large digit recognition task using MNIST data. In this setting our methods performed equally well as other approaches (or slightly better). The full details of these experiments are provided in a longer version of this work. Second, we study a visual object recognition application where some features are missing because they cannot be located in the image. Finally, we apply our methods to a problem of biological network completion, where missingness patterns of features is determined by the known structure of the network.

For all applications, we compare our iterative algorithm with five common approaches for completing missing features. **1. Zero:** Missing values were set to zero. **2. Mean:** Missing values were set to the average feature values **3. Aggregated Flags:** Features were annotated with an explicit additional feature noting whether a feature is valid or missing. To reduce the number of added features, we added a single flag for each group of features that were valid or invalid together across all instances. For example, In the vision application, all features of a landmark candidate are grouped together since they are all invalid if the match is wrong (see below). **4. kNN:** Missing features were set with the mean value obtained from the $K$ nearest neighbors instances; neighborhood was measured using a Euclidean distance in the subspace relevant to each two samples, number of neighbors was varied as $K = 3, 5, 10, 20$, and the best result is the one reported. **5. EM:** Generative model in the spirit of [4]. A Gaussian mixture model is learned by iterating between (1) learning a GMM model of the *filled* data (2) re-filling missing values using clusters means, weighted by the posterior probability that a cluster generated the sample. Covariances were assumed spherical. The number of clusters was varied as $K = 3, 5, 10, 15, 20$, and the best result is the one reported. **6. Geometric margin:** Our non-separable approach described in Sec. 3.

In all of the experiments, we used a 5-fold cross validation procedure and evaluated performance using a testing set that was not used during training. In addition, 20% of the training set was used for choosing optimization parameters, such as the kernel type, its parameters, and an early stopping point for the iterative algorithm.

### 4.1 Visual object recognition

We now consider a visual object recognition task where instances have structurally missing features. In this task we attempt to determine if an object from a certain class (automobiles) is present in a given input image. The task of classifying images based on the object class that they contain has seen much work in recent years [1, 5],and discriminative approaches have typically produced very good results [5, 12].

Features in these methods are commonly constructed from *regions of interest* (patches) in the image. These patches typically cover "landmarks" of the object, like the trunk or a headlight for a car. A typical set of patches includes several candidates for any object part,

and some images may have more candidates for a given part than others. For example, a trunk of a car may not be found in a picture of a hatch-back car, hence all its corresponding features are considered to be structurally missing from that image. Our object model contains a set of "landmarks", for which we find several matches in a given image (details are omitted due to lack of space). Fig. 2 shows examples of matches for the front windshield landmark. Because of the noisy matches, the highest scoring match often does not match the true landmark, and the number of high-quality matches (features) varies in practice. It is in precisely such a scenario that we expect our proposed algorithm to be effective.

In some cases, landmark models could provide confidence levels for each match. These could in principle be used as additional features to help the classifiers give more weight to better matches, and are expected to improve classification when the confidence measure is reliable. While this is a potentially useful approach for the current application, this paper takes a different approach: it does not use any soft confidence values but rather treats the low-confidence matches as wrong, removing them from the data.

Concretely, we located up to 10 candidate patches ($21 \times 21$ pixels) that were promising (likelihood above a given threshold) for each of the 19 landmarks in the car model. For each candidate, we compute the first 10 principal component coefficients of the image patch and concatenate these patches to form the image feature vector. If the number of patches for a given landmark is less than ten, we consider the rest to be structurally absent. We evaluated performance for this task using two levels of a 5-fold cross validation procedure as explained above. We compared several kernels and report results using the kernel that fared best on the validation set, which was usually a second order polynomial kernel.

Fig. 2c compares the accuracy of the different methods. We found the geometric approach to be significantly superior to all other methods. To further evaluate our method, we qualitatively examined the classification results for several images across the various methods. Fig. 2a shows the top 10 matches for the front windshield landmark for a representative "easy" test instance where all local features are approximately in agreement. This instance was correctly classified by all methods. In contrast, Fig. 2b shows a representative "hard" test instance where local features cluster into two different groups. In this case, the cluster of bad matches was automatically excluded yielding missing features, and our geometric approach was the only method able to classify the instance correctly.

## 4.2   Metabolic pathway reconstruction

As a final application, we consider the problem of predicting missing enzymes in metabolic pathways, a long-standing and important challenge in computational biology [15, 9]. Instances in this task have missing features due to the structure of the biochemical network. Cells use a complex network of chemical reactions to produce their chemical building blocks (Fig. 3). Each reaction transforms a set of molecular compounds (called *substrates*) into another set of molecules (*products*), and requires the presence of an *enzyme* to catalyze the reaction. It is often unknown which enzyme catalyzes a given reaction, and it is desirable to predict the identity of such missing enzymes computationally.

Our approach for predicting missing enzymes is based on the observation that enzymes in local network neighborhoods usually participate in related functions. As a result, neighboring enzyme pairs have non trivial correlations over their features that reflect their functional relations. Importantly, different types of neighborhood relations between enzyme pairs lead to different relations of their properties. For example, an enzyme in a linear chain depends on the preceding enzyme product as its substrate. Hence it is expected that the corresponding genes are co-expressed [9, 15]. On the other hand, enzymes in forking motifs (same substrate, different products) often have anti-correlated expression profiles [7].

To preserve the distinction between different neighbor relations, we defined a set of network motifs, including *forks*, *funnels* and *linear chains*. Each enzyme is represented as a vector of features that measure its relatedness to each of its neighbors. A feature vector has structurally missing entries if the enzyme does not have all types of neighbors. For example, the enzyme *PHA2* in Fig. 3 does not have a neighbor of type *fork*, and therefore all features assigned to such a neighbor are absent in the representation of the reaction *"Prephanate → Phenylpyruvate"*.

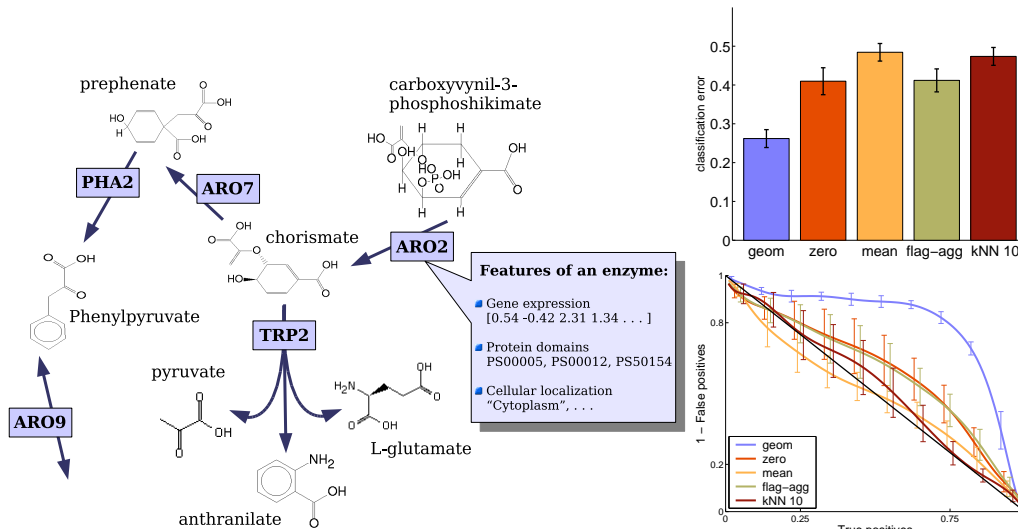

Figure 3: **Left**: A fragment of the full metabolic pathway network in *S. cerevisiae*. Chemical reactions (arrows) transform a set of molecular compounds into other compounds. Small molecules like $CO_2$ were omitted from this drawing for clarity. Reactions are catalyzed by enzymes (boxed names, e.g., *ARO7*), but in some cases these enzymes are unknown. The network imposes various neighborhood relations between enzymes assigned to reactions, like linear chains (*ARO7,PHA2*), forks (*TRP2,ARO7*) and funnels (*ARO9,PHA2*) **Top Right:** Classification accuracy for compared methods. The classification task is to identify if a candidate enzyme is in the right "neighborhood". Error bars are SEMs over 5 cross validation sets. **Bottom right:** ROC curves for the same task.

We used the metabolic network of *S. cerevisiae*, as reconstructed by Palsson and colleagues [3], after removing 14 metabolic currencies and reactions with unknown enzymes, leaving 1265 directed reactions. We used three data types: (1) A compendium of 645 gene expression experiments; each experimental condition $k$ contributed one feature, the point-wise Pearson correlation $\frac{\mathbf{x}_i(k)\mathbf{x}_j(k)}{\|\mathbf{x}_i\|\|x_j\|}$. $x_i$ is the vector of expression levels across conditions. (2) The protein-domain content of each enzyme as found by the Prosite database. Each domain $k$ contributed one feature, the point-wise symmetric $D_{KL}$ measure $\mathbf{x}_i(k)\left(\log(\mathbf{x}_i(k)/(\mathbf{x}_j(k)+\mathbf{x}_i(k))/2)\right) + \mathbf{x}_j(k)\left(\log(\mathbf{x}_j(k)/(\mathbf{x}_j(k)+\mathbf{x}_i(k))/2)\right)$. (3) The cellular localization of the protein [6]; each cellular localization contributed one feature, the point-wise Hamming distance. In total, the feature vector length was about 3900. Pathway reconstruction requires that we rank candidate enzymes by their potential to match a reaction. As a first step towards this goal, we train a binary classifier, to predict if an enzyme fits its neighborhood. We created a set of positive examples from the reactions with known enzymes ($\sim$ 520 reactions), and also created negative examples by plugging impostor genes into 'wrong' neighborhoods. We trained an SVM classifier using a 5-fold cross validation procedure as described above.

Figure 3 shows the classification error of the different methods in the gene filling task. The *geometric margin approach* achieves significantly better performance in this task. kNN achieved very poor performance compared to all other methods. One reason could be that the Euclidean distance is inappropriate for the current task and that a more elaborate distance measure needs to developed for this type of data. Learning metrics is a complicated task in general, and more so in the current problem since the feature vectors contain entries of several different types, making it unlikely that a naive distance measure would work well.

Finally, the resulting classifier is used for predicting missing enzymes, by ranking all candidate enzymes according to their match to a given neighborhood. Evaluating the quality of ranking on known enzymes (cross validation), shows that it significantly outperforms previous approaches [9] (not shown here due to space limitations). We attribute this to the ability of the current approach to preserve different types of network-neighbors as separate features in spite of creating missing values.

# 5  Discussion

We presented a novel method for max-margin training of classifiers in the presence of missing features, where the pattern of missing features is an inherent part of the domain. Instead of completing missing features as a preprocessing phase, we developed a max-margin learning objective based on a geometric interpretation of the margin when different instances essentially lie in different spaces. Using two challenging real life problems we showed that our method is significantly superior when the pattern of missing features has structure.

The standard treatment of missing features is based on the concept that missing features exist but are unobserved. This assumption underlies the approach of completing features before the data is used in classification. This paper focuses on a different scenario, in which features are inherently absent. In such cases, it is not clear why we should guess hypothetical values for undefined features, since the completed values are filled based on other observed values, and do not add information to our classifiers. In fact, by completing features that are not supposed to be part of an instance, we may be confusing the learning algorithm by presenting it with problem which may be harder than the one we actually need to solve.

Interestingly, the problem of classifying with missing features is related to another problem, where individual reliability measures are available for features at each instance separately. This is a common case in analysis scientific measurements, where the reliability of each experiment could be provided separately. For example, DNA micro-array experiments have inherent measures of experimental noise levels, and biological variability is often estimated using replicates. This problem can be viewed in the same framework described in this paper: the geometric margin must be defined separately for each instance since the different noise levels distort the relative scale of each coordinate of each instance separately. Relative to this setting, the completely missing and fully valid features discussed in this paper are extreme points on the spectrum of reliability. It will be interesting to see which aspects of the geometric formulation discussed in this paper can be extended to this new problem.

**Acknowledgement:** This paper was supported by a NSF grant DBI-0345474.

# References

[1] A. Berg, T. Berg, and J. Malik. Shape matching and object recognition using low distortion correspondence. In *CVPR*, 2005.

[2] Paul H. Calamai and Jorge J. More:9A. Projected gradient methods for linearly constrained problems. *Math. Program.*, 39(1):93–116, 1987.

[3] J. Forster, I. Famili, P. Fu, B.. Palsson, and J. Nielsen. Genome-scale reconstruction of the saccharomyces cerevisiae metabolic network. *Genome Research*, 13(2):244–253, February 2003.

[4] Z. Ghahramani and MI. Jordan. Supervised learning from incomplete data via an EM approach. In JD. Cowan, G. Tesauro, and J. Alspector, editors, *NIPS*, volume 6, pages 120–127, 1994.

[5] K. Grauman and T. Darrell. Pyramid match kernels: Discriminative classification with sets of image features. In *ICCV*, 2005.

[6] W.K. Huh, J.V. Falvo, L.C. Gerke, A.S. Carroll, R.W. Howson, J.S. Weissman, and E.K. O'Shea. Global analysis of protein localization in budding yeast. *Nature*, 425:686–691, 2003.

[7] J. Ihmels, R. Levy, and N. Barkai. Principles of transcriptional control in the metabolic network of saccharomyces cerevisiae. *Nature Biotechnology*, 22:86–92, 2003.

[8] D. Jensen and J. Neville. Linkage and autocorrelation cause feature selection bias in relational learning. In *ICML*, 2002.

[9] P. Kharchenko, D. Vitkup, and GM. Church. Filling gaps in a metabolic network using expression information. *Bioinformatics*, 20:I178–I185, 2003.

[10] R.J.A. Little and D.B. Rubin. *Statistical Analysis with Missing Data*. NY wiley, 1987.

[11] MS. Lobo, L. Vandenberghe, S. Boyd, and H. Lebret. Applications of second-order cone programming. *Linear Algebra and its Applications*, 284:193–228, 1998.

[12] A. Quattoni, M. Collins, and T. Darrell. Conditional random fields for object recognition. In LK. Saul, Y. Weiss, and L. Bottou, editors, *NIPS 17*, pages 1097–1104, 2005.

[13] B. Schölkopf and A.J. Smola. *Learning with Kernels: Support Vector Machines, Regularization Optimization and Beyond*. MIT Press, Cambridge, MA, 2002.

[14] V.N. Vapnik. *The nature of statistical learning theory*. SpringerVerlag, 1995.

[15] J. P. Vert and Y. Yamanishi. Supervised graph inference. In LK. Saul, Y. Weiss, and L. Bottou, editors, *NIPS 17*, pages 1433–1440, 2004.
